# Code-specific policy gradient rules for spiking neurons

**Henning Sprekeler**[*]    **Guillaume Hennequin**    **Wulfram Gerstner**
Laboratory for Computational Neuroscience
École Polytechnique Fédérale de Lausanne
1015 Lausanne

## Abstract

Although it is widely believed that reinforcement learning is a suitable tool for describing behavioral learning, the mechanisms by which it can be implemented in networks of spiking neurons are not fully understood. Here, we show that different learning rules emerge from a policy gradient approach depending on which features of the spike trains are assumed to influence the reward signals, i.e., depending on which neural code is in effect. We use the framework of Williams (1992) to derive learning rules for arbitrary neural codes. For illustration, we present policy-gradient rules for three different example codes - a spike count code, a spike timing code and the most general "full spike train" code - and test them on simple model problems. In addition to classical synaptic learning, we derive learning rules for intrinsic parameters that control the excitability of the neuron. The spike count learning rule has structural similarities with established Bienenstock-Cooper-Munro rules. If the distribution of the relevant spike train features belongs to the natural exponential family, the learning rules have a characteristic shape that raises interesting prediction problems.

## 1   Introduction

Neural implementations of reinforcement learning have to solve two basic credit assignment problems: (a) the temporal credit assignment problem, i.e., the question which of the actions that were taken in the past were crucial to receiving a reward later and (b) the spatial credit assignment problem, i.e., the question, which neurons in a population were important for getting the reward and which ones were not.

Here, we argue that an additional credit assignment problem arises in implementations of reinforcement learning with spiking neurons. Presume that we know that the spike pattern of one specific neuron within one specific time interval was crucial for getting the reward (that is, we have already solved the first two credit assignment problems). Then, there is still one question that remains: Which *feature* of the spike pattern was important for the reward? Would any spike train with the same number of spikes yield the same reward or do we need precisely timed spikes to get it? This credit assignment problem is in essence the question which neural code the output neuron is (or should be) using. It becomes particularly important, if we want to change neuronal parameters like synaptic weights in order to maximize the likelihood of getting the reward again in the future. If only the spike count is relevant, it might not be very effective to spend a lot of time and energy on the difficult task of learning precisely timed spikes.

The most modest and probably most versatile way of solving this problem is not to make any assumption on the neural code but to assume that *all* features of the spike train were important. In

---

[*]E-Mail: `henning.sprekeler@epfl.ch`

this case, neuronal parameters are changed such that the likelihood of repeating exactly the same spike train for the same synaptic input is maximized. This approach leads to a learning rule that was derived in a number of recent publications [3, 5, 13]. Here, we show that a whole class of learning rules emerges when prior knowledge about the neural code at hand is available. Using a policy-gradient framework, we derive learning rules for neural parameters like synaptic weights or threshold parameters that maximize the expected reward.

Our aims are to (a) develop a systematic framework that allows to derive learning rules for arbitrary neural parameters for different neural codes, (b) provide an intuitive understanding how the resulting learning rules work, (c) derive and test learning rules for specific example codes and (d) to provide a theoretical basis why code-specific learning rules should be superior to general-purpose rules. Finally, we argue that the learning rules contain two types of prediction problems, one related to reward prediction, the other to response prediction.

## 2 General framework

### 2.1 Coding features and the policy-gradient approach

The basic setup is the following: let there be a set of different input spike trains $X^\mu$ to a single post-synaptic neuron, which in response generates stochastic output spike trains $Y^\mu$. In the language of partially observable Markov decision processes, the input spike trains are observations that provide information about the state of the animal and the output spike trains are controls that influence the action choice. Depending on both of these spike trains, the system receives a reward. The goal is to adjust a set of parameters $\theta_i$ of the postsynaptic neuron such that it maximizes the expectation value of the reward.

Our central assumption is that the reward $R$ does not depend on the full output spike train, but only on a set of *coding features* $F_j(Y)$ of the output spike train: $R = R(\mathbf{F}, X)$. Which coding features $\mathbf{F}$ the reward depends on is in fact a choice of a neural code, because all other features of the spike train are not behaviorally relevant. Note that there is a conceptual difference to the notion of a neural code in sensory processing, where the coding features convey information about input signals, not about the output signal or rewards.

The expectation value of the reward is given by $\langle R \rangle = \sum_{\mathbf{F},X} R(\mathbf{F}, X) P(\mathbf{F}|X, \theta) P(X)$, where $P(X)$ denotes the probability of the presynaptic spike trains and $P(\mathbf{F}|X, \theta)$ the conditional probability of generating the coding feature $\mathbf{F}$ given the input spike train $X$ and the neuronal parameters $\theta$. Note that the only component that explicitly depends on the neural parameters $\theta_i$ is the conditional probability $P(\mathbf{F}|X, \theta)$. The reward is conditionally independent of the neural parameters $\theta_i$ given the coding feature $\mathbf{F}$. Therefore, if we want to optimize the expected reward by employing a gradient ascent method, we get a learning rule of the form

$$\partial_t \theta_i = \eta \sum_{\mathbf{F},X} R(\mathbf{F}, X) P(X) \partial_{\theta_i} P(\mathbf{F}|X, \theta) \tag{1}$$

$$= \eta \sum_{\mathbf{F},X} P(X) P(\mathbf{F}|X, \theta) R(\mathbf{F}, X) \partial_{\theta_i} \ln P(\mathbf{F}|X, \theta) . \tag{2}$$

If we choose a small learning rate $\eta$, the average over presynaptic patterns $X$ and coding features $F$ can be replaced by a time average. A corresponding online learning rule therefore results from dropping the average over $X$ and $F$:

$$\partial_t \theta_i = \eta R(\mathbf{F}, X) \partial_{\theta_i} \ln P(\mathbf{F}|X, \theta) . \tag{3}$$

This general form of learning rule is well known in policy-gradient approaches to reinforcement learning [1, 12].

### 2.2 Learning rules for exponentially distributed coding features

The joint distribution of the coding features $F_j$ can always be factorized into a set of conditional distributions $P(\mathbf{F}|X) = \prod_i P(F_i|X; F_1, ..., F_{i-1})$. We now make the assumption that the conditional distributions belong to the natural exponential family (NEF): $P(F_i|X; F_1, ..., F_{i-1}, \theta) =$

$h(F_i)\exp(C_iF_i - A(C_i))$, where the $C_i$ are parameters that depend on the input spike train $X$, the coding features $F_1, ..., F_{i-1}$ and the neural parameters $\theta_i$. $h(F_i)$ is a function of $F_i$ and $A_i(C_i)$ is function that is characteristic for the distribution and depends only on the parameters $C_i$. Note that the NEF is a relatively rich class of distributions, which includes many canonical distributions like the Poisson, Bernoulli and the Gaussian distribution (the latter with fixed variance).

Under these assumptions, the learning rule (3) takes a characteristic shape:

$$\partial_t \theta_i = \eta R(\mathbf{F}, X) \sum_j \frac{F_j - \mu_j}{\sigma_j^2} \partial_{\theta_i} \mu_j \, , \qquad (4)$$

where $\mu_i$ and $\sigma_i^2$ are the mean and the variance of the conditional distribution $P(F_i | X, F_1, ..., F_{i-1}, \theta)$ and therefore also depend on the input $X$, the coding features $F_1, ..., F_{i-1}$ and the parameters $\theta$. Note that correlations between the coding features are implicitly accounted for by the dependence of $\mu_i$ and $\sigma_i$ on the other features. The summation over different coding features arises from the factorization of the distribution, while the specific shape of the summands relies on the assumption of normal exponential distributions [for a proof, cf. 12].

There is a simple intuition why the learning rule (4) performs gradient ascent on the mean reward. The term $F_j - \mu_j$ fluctuates around zero on a trial-to-trial basis. If these fluctuations are positively correlated with the trial fluctuations of the reward $R$, i.e., $\langle R(F_j - \mu_j) \rangle > 0$, higher values of $F_j$ lead to higher reward, so that the mean of the coding feature should be increased. This increase is implemented by the term $\partial_{\theta_i} \mu_j$, which changes the neural parameter $\theta_i$ such that $\mu_j$ increases.

## 3 Examples for Coding Features

In this section, we illustrate the framework by deriving policy-gradient rules for different neural codes and show that they can solve simple computational tasks.

The neuron type we are using is a simple Poisson-type neuron model where the postsynaptic firing rate is given by a nonlinear function $\rho(u)$ of the membrane potential $u$. The membrane potential $u$, in turn, is given by the sum of the EPSPs that are evoked by the presynaptic spikes, weighted with the respective synaptic weights:

$$u(t) = \sum_{i,f} w_i \epsilon(t - t_i^f) \, , =: \sum_i w_i \mathrm{PSP}_i(t) \, , \qquad (5)$$

where $t_i^f$ denote the time of the $f$-th spike in the $i$-th presynaptic neuron. $\epsilon(t - t_i^f)$ denotes the shape of the postsynaptic potential evoked by a single presynaptic spike at time $t_i^f$. For future use, we have introduced $\mathrm{PSP}_i$ as the postsynaptic potential that would be evoked by the $i$-th presynaptic spike train alone, if the synaptic weight were unity.

The parameters that one could optimize in this neuron model are (a) the synaptic weights and (b) parameters in the dependence of the firing rate $\rho$ on the membrane potential. The first case is the standard case of synaptic plasticity, the second corresponds to a reward-driven version of intrinsic plasticity [cf. 10].

### 3.1 Spike Count Codes: Synaptic plasticity

Let us first assume that the coding feature is the number $N$ of spikes within a given time window $[0, T]$ and that the reward is delivered at the end of this period. The probability distribution for the spike count is a Poisson distribution $P(N) = \mu^N \exp(-\mu)/N!$ with a mean $\mu$ that is given by the integral of the firing rate $\rho$ over the interval $[0, T]$:

$$\mu = \int_0^T \rho(t') \, \mathrm{d}t' \, . \qquad (6)$$

The dependence of the distribution $P(N)$ on the presynaptic spike trains $X$ and the synaptic weights $w_i$ is hidden in the mean spike count $\mu$, which naturally depends on those factors through the postsynaptic firing rate $\rho$.

Because the Poisson distribution belongs to the NEF, we can derive a synaptic learning rule by using equation (4) and calculating the particular form of the term $\partial_{w_i}\mu$:

$$\partial_t w_i = \eta R \frac{N-\mu}{\mu} \int_0^T [\partial_u \rho](t') \mathrm{PSP}_i(t') \, \mathrm{d}t' \,. \tag{7}$$

This learning rule has structural similarities with the Bienenstock-Cooper-Munro (BCM) rule [2]: The integral term has the structure of an eligibility trace that is driven by a simple Hebbian learning rule. In addition, learning is modulated by a factor that compares the current spike count ("rate") with the expected spike count ("sliding threshold" in BCM theory). Interestingly, the functional role of this factor is very different from the one in the original BCM rule: It is not meant to introduce selectivity [2], but rather to exploit trial fluctuations around the mean spike count to explore the structure of the reward landscape.

We test the learning rule on a 2-armed bandit task (Figure 1A). An agent has the choice between two actions. Depending on which of two states the agent is in, action $a_1$ or action $a_2$ is rewarded ($R = 1$), while the other action is punished ($R = -1$). The state information is encoded in the rate pattern of 100 presynaptic neurons. For each state, a different input pattern is generated by drawing the firing rate of each input neuron independently from an exponential distribution with a mean of 10Hz. In each trial, the input spike trains are generated anew from Poisson processes with these neuron- and state-specific rates. The agent chooses its action stochastically with probabilities that are proportional to the spike counts of two output neurons: $p(a_k|s) = N_k/(N_1 + N_2)$. Because the spike counts depend on the state via the presynaptic firing rates, the agent can choose different actions for different states. Figure 1B and C show that the learning rule learns the task by suppressing activity in the neuron that encodes the punished action.

In all simulations throughout the paper, the postsynaptic neurons have an exponential rate function $g(u) = \exp(\gamma(u - u_0))$, where the threshold is $u_0 = 1$. The sharpness parameter $\gamma$ is set to either $\gamma = 1$ (for the 2-armed bandit task) or $\gamma = 3$ (for the spike latency task). Moreover, the postsynaptic neurons have a membrane potential reset after each spike (i.e., relative refractoriness), so that the assumption of a Poisson distribution for the spike counts is not necessarily fulfilled. It is worth noting that this did not have an impeding effect on learning performance.

## 3.2 Spike Count Codes: Intrinsic plasticity

Let us now assume that the rate of the neuron is given by a function $\rho(u) = g(\gamma(u - u_0))$ which depends on the *threshold parameters* $u_0$ and $\gamma$. Typical choices for the function $g$ would be an exponential (as used in the simulations), a sigmoid or a threshold linear function $g(x) = \ln(1 + \exp(x))$.

By intrinsic plasticity we mean that the parameters $u_0$ and $\gamma$ are learned instead of or in addition to the synaptic weights. The learning rules for these parameters are essentially the same as for the synaptic weights, only that the derivative of the mean spike count is taken with respect to $u_0$ and $\gamma$, respectively:

$$\partial_t u_0 \;=\; \eta \frac{N-\mu}{\mu} \partial_{u_0}\mu = -\eta \frac{N-\mu}{\mu} \int_0^T \gamma g'(\gamma(u(t) - u_0)) \, \mathrm{d}t \tag{8}$$

$$\partial_t \gamma \;=\; \eta \frac{N-\mu}{\mu} \partial_\gamma \mu = \eta \frac{N-\mu}{\mu} \int_0^T g'(\gamma(u(t) - u_0))(u(t) - u_0) \, \mathrm{d}t \,. \tag{9}$$

Here, $g' = \partial_x g(x)$ denotes the derivative of the rate function $g$ with respect to its argument.

## 3.3 First Spike-Latency Code: Synaptic plasticity

As a second coding scheme, let us assume that the reward depends only on the latency $\hat{t}$ of the first spike after stimulus onset. More precisely, we assume that each trial starts with the onset of the presynaptic spike trains $X$ and that a reward is delivered at the time of the first spike. The reward depends on the latency of that spike, so that certain latencies are favored.

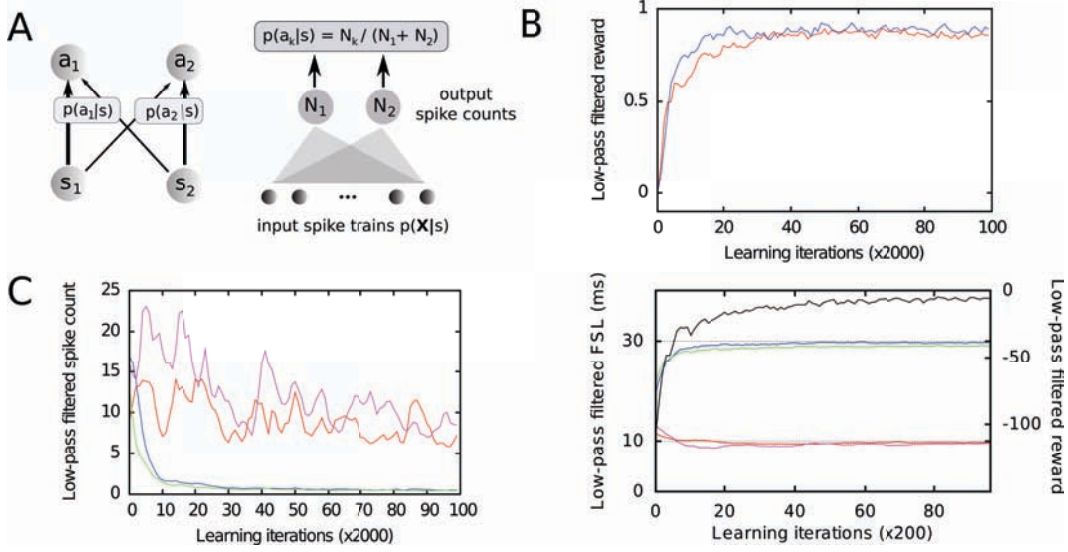

Figure 1: Simulations for code-specific learning rules. **A** 2-armed bandit task: The agent has to choose among two actions $a_1$ and $a_2$. Depending on the state ($s_1$ or $s_2$), a different action is rewarded (thick arrows). The input states are modelled by different firing rate patterns of the input neurons. The probability of choosing the actions is proportional to the spike counts of two output neurons: $p(a_k|s) = N_k/(N_1 + N_2)$. **B** Learning curves of the 2-armed bandit. Blue: Spike count learning rule (7), Red: Full spike train rule (16). **C** Evolution of the spike count in response to the two input states during learning. Both rewards (panel B) and spike counts (panel C) are low-pass filtered with a time constant of 4000 trials. **D** Learning of first spike latencies with the latency rule (11). Two different output neurons are to learn to fire their first spike at given target latencies $L_{1,2}$. We present one of two fixed input spike train patterns ("stimuli") to the neurons in randomly interleaved trials. The input spike train for each input neuron is drawn separately for each stimulus by sampling once from a Poisson process with a rate of 10Hz. Reward is given by the negative squared difference between the target latency (stimulus 1: $L_1 = 10$ms, $L_2 = 30$ms, stimulus 2: $L_1 = 30$ms, $L_2 = 10$ms) and the actual latency of the trial, summed over the two neurons. The colored curves show that the first spike latencies of neurons 1 (green, red) and neuron 2 (purple, blue) converge to the target latencies. The black curve (scale on the right axis) shows the evolution of the reward during learning.

The probability distribution of the spike latency is given by the product of the firing probability at time $t$ and the probability that the neuron did not fire earlier:

$$P(t) = \rho(t) \exp\left(-\int_0^t \rho(t')\,\mathrm{d}t'\right) \tag{10}$$

Using eq. (3) for this particular distribution, we get the synaptic learning rule:

$$\Delta_t w_i = \eta R \left(\frac{[\rho_u]'(t)\mathrm{PSP}_i(t)}{\rho(t)} - \int_0^t [\rho_u]'(t')\mathrm{PSP}_i(t')\,\mathrm{d}t'\right) \tag{11}$$

In Figure 1D, we show that this learning rule can learn to adjust the weights of two neurons such that their first spike latencies approximate a set of target latencies.

### 3.4 The Full Spike Train Code: Synaptic plasticity

Finally, let us consider the most general coding feature, namely, the full spike train. Let us start with a time-discretized version of the spike train with a discretization that is sufficiently narrow to allow at most one spike per time bin. In each time bin $[t, t+\Delta t]$, the number of spikes $Y_t$ follows a Bernoulli distribution with spiking probability $p_t$, which depends on the input and on the recent history of the neuron. Because the Bernoulli distribution belongs to the NEF, the associated policy-gradient rule can be derived using equation (4):

$$\Delta_t w_i = \eta R \sum_t \frac{Y_t - p_t}{p_t(1 - p_t)} \partial_{w_i} p_t \tag{12}$$

The firing probability $p_t$ depends on the instantaneous firing rate $\rho_t$: $p_t = 1 - \exp(-\rho_t \Delta t)$, yielding:

$$\partial_t w_i \;=\; \eta R \sum_t \frac{Y_t - p_t}{p_t(1-p_t)} \; \underbrace{[\partial_\rho p_t]}_{=\Delta t(1-p_t)} \; [\partial_{w_i}\rho_t] \tag{13}$$

$$=\; \eta R \sum_t (Y_t - p_t) \frac{\partial_u \rho_t}{p_t} \mathrm{PSP}_i(t)\Delta t \tag{14}$$

This is the rule that should be used in discretized simulations. In the limit $\Delta t \to 0$, $p_t$ can be approximated by $p_t \to \rho \Delta t$, which leads to the continuous time version of the rule:

$$\partial_t w_i \;=\; \eta R \lim_{t\to 0} \sum_t \left( \frac{Y_t}{\Delta t} - \rho_t \right) \frac{\partial_u \rho_t}{\rho_t} \mathrm{PSP}_i(t)\Delta t \tag{15}$$

$$=\; \eta R \int (Y(t) - \rho(t)) \frac{[\partial_u \rho](t)}{\rho(t)} \mathrm{PSP}_i(t)\,\mathrm{d}t\,. \tag{16}$$

Here, $Y(t) = \sum_{t_i} \delta(t - t_i)$ is now a sum of $\delta$-functions. Note that the learning rule (16) was already proposed by Xie and Seung [13] and Florian [3] and, slightly modified for supervised learning, by Pfister et al. [5].

Following the same line, policy gradient rules can also be derived for the intrinsic parameters of the neuron, i.e., its threshold parameters (see also [3]).

## 4   Why use code-specific rules when more general rules are available?

Obviously, the learning rule (16) is the most general in the sense that it considers the whole spike train as a coding feature. All possible other features are therefore captured in this learning rule. The natural question is then: what is the advantage of using rules that are specialized for one specific code?

Say, we have a learning rule for two coding features $F_1$ and $F_2$, of which only $F_1$ is correlated with reward. The learning rule for a particular neuronal parameter $\theta$ then has the following structure:

$$\partial_t \theta \;=\; \eta R(F_1)\left( \frac{(F_1 - \mu_1)}{\sigma_1^2}\frac{\partial \mu_1}{\partial \theta} + \frac{F_2 - \mu_2}{\sigma_2^2}\frac{\partial \mu_2}{\partial \theta} \right) \tag{17}$$

$$\approx\; \eta \left( R(\mu_1) + \left.\frac{\partial R}{\partial F_1}\right|_{\mu_1}(F_1 - \mu_1) \right) \left( \frac{F_1 - \mu_1}{\sigma_1^2}\frac{\partial \mu_1}{\partial \theta} + \frac{F_2 - \mu_2}{\sigma_2^2}\frac{\partial \mu_2}{\partial \theta} \right) \tag{18}$$

$$=\; \eta \left.\frac{\partial R}{\partial F_1}\right|_{\mu_1}\frac{(F_1 - \mu_1)^2}{\sigma_1^2}\frac{\partial \mu_1}{\partial \theta} + \eta \left.\frac{\partial R}{\partial F_1}\right|_{\mu_1}\frac{(F_1 - \mu_1)(F_2 - \mu_2)}{\sigma_2^2}\frac{\partial \mu_2}{\partial \theta} \tag{19}$$

$$+\eta R(\mu_1)\frac{F_1 - \mu_1}{\sigma_1^2}\frac{\partial \mu_1}{\partial \theta} \qquad +\eta R(\mu_1)\frac{F_2 - \mu_2}{\sigma_2^2}\frac{\partial \mu_2}{\partial \theta} \tag{20}$$

Of the four terms in lines (19-20), only the first term has non-vanishing mean when taking the trial average. The other terms are simply noise and therefore more hindrance than help when trying to maximize the reward. When using the full learning rule for both features, the learning rate needs to be decreased until an agreeable signal-to-noise ratio between the drift introduced by the first term and the diffusion caused by the other terms is reached. Therefore, it is desirable for faster learning to reduce the effects of these noise terms. This can be done in two ways:

- The terms in eq. (20) can be reduced by reducing $R(\mu_1)$. This can be achieved by subtracting a suitable reward baseline from the current reward. Ideally, this should be done in a stimulus-specific way (because $\mu_1$ depends on the stimulus), which leads to the notion of a reward prediction error instead of a pure reward signal. This approach is in line with both standard reinforcement learning theory [4] and the proposal that neuromodulatory signals like dopamine represent reward prediction error instead of reward alone.

- The term in eq. (20) can be removed by skipping those terms in the original learning that are related to coding feature $F_2$. This corresponds to using the learning rule for those features that are in fact correlated with reward while suppressing those that are not correlated with reward. The central argument for using code-specific learning rules is therefore the signal-to-noise ratio. In extreme cases, where a very general rule is used for a very specific task, a very large number of coding dimensions may merely give rise to noise in the learning dynamics, while only one is relevant and causes systematic changes.

These considerations suggest that the spike count rule (7) should outperform the full spike train rule (16) in tasks where the reward is based purely on spike count. Unfortunately, we could not yet substantiate this claim in simulations. As seen in Figure 1B, the performance of the two rules is very similar in the 2-armed bandit task. This might be due to a noise bottleneck effect: there are several sources of noise in the learning process, the strongest of which limits the performance. Unless the "code-specific noise" is dominant, code-specific learning rules will have about the same performance as general purpose rules.

## 5 Inherent Prediction Problems

As shown in section 4, the policy-gradient rule with a reduced amount of noise in the gradient estimate is one that takes only the relevant coding features into account and subtracts the trial mean of the reward:

$$\partial_t \theta = \eta(R - R(\mu_1, \mu_2, ...)) \sum_j \frac{F_j - \mu_j}{\sigma_j^2} \partial_\theta \mu_j \tag{21}$$

This learning rule has a conceptually interesting structure: Learning takes place only when two conditions are fulfilled: the animal did something unexpected ($F_j - \mu_i$) and receives an unexpected reward ($R - R(\mu_1, \mu_2, ...)$). Moreover, it raises two interesting prediction problems: (a) the prediction of the trial average $\mu_j$ of the coding feature conditioned on the stimulus and (b) the reward that is expected if the coding feature takes its mean value.

### 5.1 Prediction of the coding feature

In the cases where we could derive the learning rule analytically, the trial average of the coding feature could be calculated from intrinsic properties of the neuron like its membrane potential. Unfortunately, it is not clear a priori that the information necessary for calculating this mean is always available. This should be particularly problematic when trying to extend the framework to coding features of populations, where the population would need to know, e.g., membrane properties of its members.

An interesting alternative is that the trial mean is calculated by a prediction system, e.g., by top-down signals that use prior information or an internal world model to predict the expected value of the coding feature. Learning would in this case be modulated by the mismatch of a top-down prediction of the coding feature - represented by $\mu_j(X)$ - and the real value of $F_j$, which is calculated by a "bottom-up" approach. This interpretation bears interesting parallels to certain approaches in sensory coding, where the interpretation of sensory information is based on a comparison of the sensory input with an internally generated prediction from a generative model [cf. 6]. There is also some experimental evidence for neural stimulus prediction even in comparably low-level systems as the retina [e.g. 8].

Another prediction system for the expected response could be a population coding scheme, in which a population of neurons is receiving the same input and should produce the same output. Any neuron of the population could receive the average population activity as a prediction of its own mean response. It would be interesting to study the relation of such an approach with the one recently proposed for reinforcement learning in populations of spiking neurons [11].

### 5.2 Reward prediction

The other quantity that should be predicted in the learning rule is the reward one would get when the coding feature would take the value of its mean. If the distribution of the coding feature is

sufficiently narrow so that in the range $F$ takes for a given stimulus, the reward can be approximated by a linear function, the reward $R(\mu)$ at the mean is simply the expectation value of the reward given the stimulus:

$$R(\mu) \approx \langle R(F) \rangle_{F|X} \tag{22}$$

The relevant quantity for learning is therefore a reward prediction error $R(F) - \langle R(F) \rangle_{F|X}$. In classical reinforcement learning, this term is often calculated in an actor-critic architecture, where some external module - the critic - learns the expected future reward either for states alone or for state-action pairs. These values are then used to calculate the expected reward for the current state or state-action pair. The difference between the reward that was really received and the predicted reward is then used as a reward prediction error that drives learning. There is evidence that dopamine signals in the brain encode prediction error rather than reward alone [7].

## 6 Discussion

We have presented a general framework for deriving policy-gradient rules for spiking neurons and shown that different learning rules emerge depending on which features of the spike trains are assumed to influence the reward signals. Theoretical arguments suggest that code-specific learning rules should be superior to more general rules, because the noise in the estimate of the gradient should be smaller. More simulations will be necessary to check if this is indeed the case and in which applications code-specific learning rules are advantageous.

For exponentially distributed coding features, the learning rule has a characteristic structure, which allows a simple intuitive interpretation. Moreover, this structure raises two prediction problems, which may provide links to other concepts: (a) the notion of using a reward prediction error to reduce the variance in the estimate of the gradient creates a link to actor-critic architectures [9] and (b) the notion of coding feature prediction is reminiscent of combined top-down–bottom-up approaches, where sensory learning is driven by the mismatch of internal predictions and the sensory signal [6].

The fact that there is a whole class of code-specific policy-gradient learning rules opens the interesting possibility that neuronal learning rules could be controlled by metalearning processes that shape the learning rule according to what neural code is in effect. From the biological perspective, it would be interesting to compare spike-based synaptic plasticity in different brain regions that are thought to use different neural codes and see if there are systematic differences.

## References

[1] Baxter, J. and Bartlett, P. (2001). Infinite-horizon policy-gradient estimation. *Journal of Artificial Intelligence Research*, 15(4):319–350.

[2] Bienenstock, E., Cooper, L., and Munroe, P. (1982). Theory of the development of neuron selectivity: orientation specificity and binocular interaction in visual cortex. *Journal of Neuroscience*, 2:32–48. reprinted in Anderson and Rosenfeld, 1990.

[3] Florian, R. V. (2007). Reinforcement learning through modulation of spike-timing-dependent synaptic plasticity. *Neural Computation*, 19:1468–1502.

[4] Greensmith, E., Bartlett, P., and Baxter, J. (2004). Variance reduction techniques for gradient estimates in reinforcement learning. *The Journal of Machine Learning Research*, 5:1471–1530.

[5] Pfister, J.-P., Toyoizumi, T., Barber, D., and Gerstner, W. (2006). Optimal spike-timing dependent plasticity for precise action potential firing in supervised learning. *Neural Computation*, 18:1309–1339.

[6] Rao, R. P. and Ballard, D. H. (1999). Predictive coding in the visual cortex: A functional interpretation of some extra-classical receptive-field effects. *Nature Neuroscience*, 2(1):79–87.

[7] Schultz, W., Dayan, P., and Montague, R. (1997). A neural substrate for prediction and reward. *Science*, 275:1593–1599.

[8] Schwartz, G., Harris, R., Shrom, D., and II, M. (2007). Detection and prediction of periodic patterns by the retina. *Nature Neuroscience*, 10:552–554.

[9] Sutton, R. and Barto, A. (1998). *Reinforcement learning*. MIT Press, Cambridge.

[10] Triesch, J. (2007). Synergies between intrinsic and synaptic plasticity mechanisms. *Neural computation*, 19:885 –909.

[11] Urbanczik, R. and Senn, W. (2009). Reinforcement learning in populations of spiking neurons. *Nat Neurosci*, 12(3):250–252.

[12] Williams, R. (1992). Simple statistical gradient-following methods for connectionist reinforcement learning. *Machine Learning*, 8:229–256.

[13] Xie, X. and Seung, H. (2004). Learning in neural networks by reinforcement of irregular spiking. *Physical Review E*, 69(4):41909.

